# A PASSIVE SHARED ELEMENT ANALOG ELECTRICAL COCHLEA

Joe Eisenberg
Bioeng. Group
U.C. Berkeley

David Feld
Dept. Elect. Eng.
207-30 Cory Hall
U.C. Berkeley
Berkeley, CA. 94720

Edwin Lewis
Dept. Elect. Eng.
U.C. Berkeley

## ABSTRACT

We present a simplified model of the micromechanics of the human cochlea, realized with electrical elements. Simulation of the model shows that it retains four signal processing features whose importance we argue on the basis of engineering logic and evolutionary evidence. Furthermore, just as the cochlea does, the model achieves massively parallel signal processing in a structurally economic way, by means of shared elements. By extracting what we believe are the five essential features of the cochlea, we hope to design a useful front-end filter to process acoustic images and to obtain a better understanding of the auditory system.

## INTRODUCTION

Results of psychoacoustical and physiological experiments in humans indicate that the auditory system creates acoustic images via massively parallel neural computations. These computations enable the brain to perform voice detection, sound localization, and many other complex tasks. For example, by recording a random signal with a wide range of frequency components, and playing this signal simultaneously through both channels of a stereo headset, one causes the brain to create an acoustical image of a "shsh" sound in the center of the head. Delaying the presentation of just one frequency component in the random signal going to one ear and simultaneously playing the original signal to the other ear, one would still have the image of a "shsh" in the center of the head; however if one mentally searches the acoustical image space carefully, a clear tone can be found far off to one side of the head. The frequency of this tone will be that of the component with the time delay to one ear. Both ears still are receiving wide-band random signals. The isolated tone will not be perceptible from the signal to either ear alone; but with both signals together, the brain has enough data to isolate the delayed tone in an acoustical image. The brain achieves this by massively parallel neural computation.

Because the acoustic front-end filter for the brain is the cochlea, people have proposed that analogs of the cochlea might serve well as front-end filters for man-made processors of acoustical images (Lyon, Mead, 1988). If we were to base a cochlear analog on current biophysical models of this structure, it would be extraordinarily complicated and extremely difficult to realize with hardware. Because of this, we want to start with a cochlear model that incorporates a minimum set of essential ingredients. The ears of lower vertebrates, such as alligators and frogs, provide some clues to help identify those ingredients. These animals presumably have to compute acoustic images similar to ours,

but they do not have cochleas. The acoustic front-end filters in the ears of these animals evolved independently and in parallel to the evolution of the cochlea. Nevertheless, those front-end filters share four functional properties with the part of the cochlea which responds to the lower 7 out of 10 octaves of hearing (20 Hz. to 2560 Hz.):

1. They are multichannel filters with each channel covering a different part of the frequency spectrum.
2. Each channel is a relatively broad-band frequency filter.
3. Each filter has an extremely steep high-frequency band edge (typically 60 to 200 db/oct).
4. Each filter has nearly linear phase shift as a function of frequency, within its passband.

The front-end acoustical filters of lower vertebrates also have at least one structural feature in common with the cochlea: namely, the various filter channels share their dynamic components. This is the fifth property we choose to include. Properties 1 and 3 provide good resolution in frequency; properties 2 and 4 are what filter designers would add to provide good resolution in time.

In order to compute acoustical images with the neural networks in our brain, we need both kinds of resolution: time and frequency. Shared elements, a structural feature, has obvious advantages for economy of construction. The fact that evolution has come to these same front-end filter properties repeatedly suggests that these properties have compelling advantages with respect to an animal's survival. We submit that we can realize all of these properties very well with the simplest of the modern cochlear models, namely that of Joseph Zwislocki (1965). This is a transmission line model made entirely of passive elements.

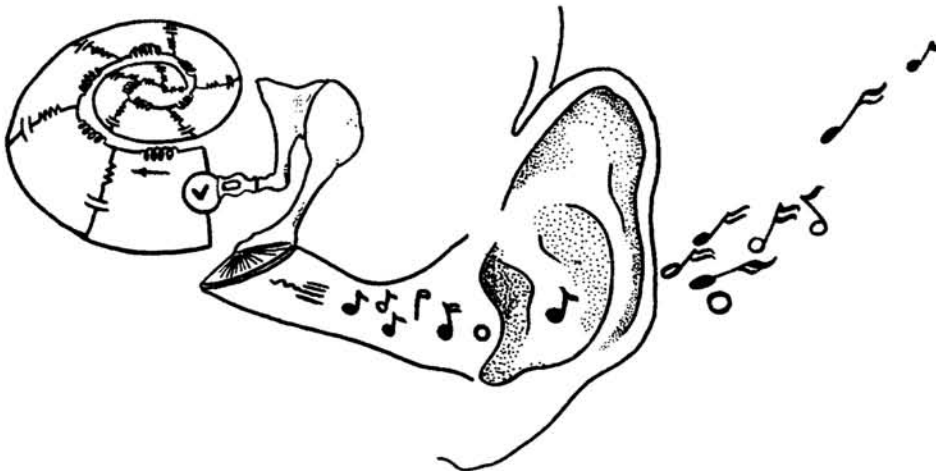

Figure 1 -    Drawing of the ear with the cochlea represented by an electrical analog.

# A COCHLEAR MODEL

In order to illustrate Zwislocki's model, a quick review of the mechanics of the cochlea is useful. Figure 1 depicts the ear with the cochlea represented by an electrical analog. A sound pressure wave enters the outer ear and strikes the ear drum which, in turn, causes the three bones of the middle ear to vibrate. The last bone, known as the stapes, is connected to the oval window (represented in figure 1 by the voltage source at the beginning of the electrical analog), where the acoustic energy enters the cochlea. As the acoustic energy is transferred to the oval window, a fluid-mechanical wave is formed along a structure known as the basilar membrane (This membrane and the surrounding fluid is represented by the series and shunt circuit elements of figure 1). As the basilar membrane vibrates, the acoustical signal is transduced to neural impulses which travel along the auditory nerve, carrying the data used by the nervous system to compute auditory images. Figure 2 is taken from a paper by Zweig et al. (1976), and depicts an uncoiled cochlea. As the fluid-mechanical wave travels through the cochlea : 1) The wave gradually slows down, and 2) The higher-frequency components of the wave are absorbed, leaving an increasingly narrower band of low-frequency components proceeding on toward the far end of the cochlea. If we were to uncoil and enlarge the basilar membrane it would look like a swim fin (figure 3). If we now were to push on the basilar membrane, it would push back like a spring. It is most compliant at the wide, thin end of the fin. Thus as one moves along the basilar membrane from its basal to apical end, its compliance increases. Zwislocki's transmission-line model was tapered in this same way.

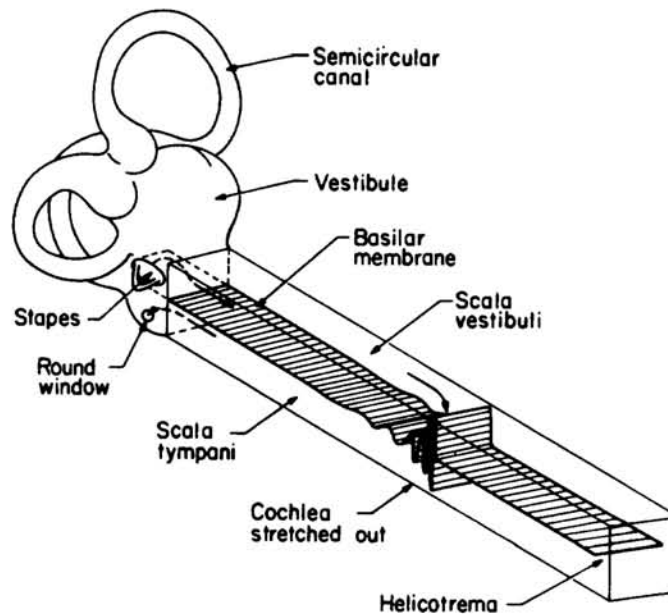

Figure 2 - Uncoiled cochlea (Zweig, 1976).

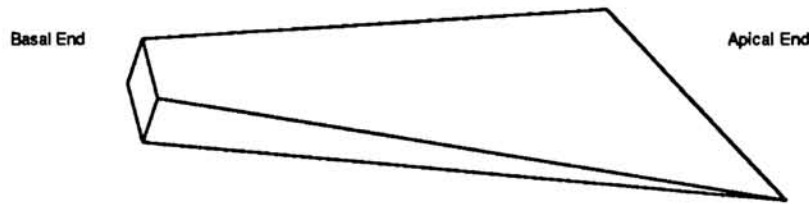

Figure 3 -    Simplified uncoiled and enlarged drawing of the basilar
              membrane.

Zwislocki's model of the cochlea is a distributed parameter transmission line. Figure 4 shows a lumped electrical analog of the model. The series elements ($L_1,...L_n$) represent the local inertia of the cochlear fluid. The shunt capacitive elements ($C_1,...C_n$) represent the local compliance of the basilar membrane. The shunt resistive elements ($R_1,...R_n$) represent the local viscous resistance of the basilar membrane and associated fluid. The model has one input and a huge number of outputs. The input, sound pressure at the oval window, is represented here as a voltage source. The outputs are either the displacements or the velocities of the various regions of the basilar membrane.

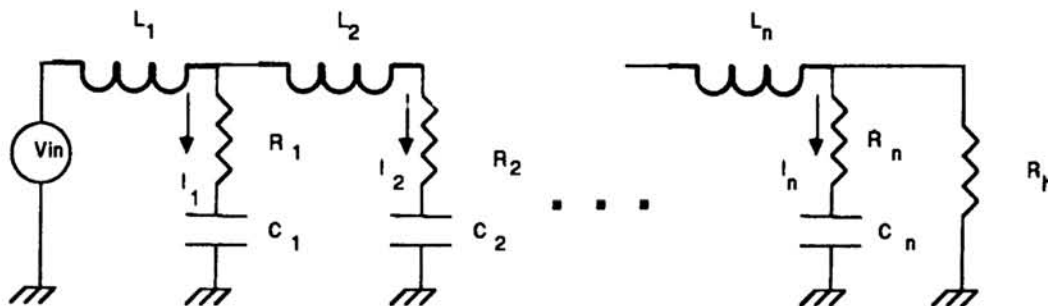

Figure 4 -    Transmission line model of the cochlea represented as an
              electrical circuit.

In the electrical analog, shown in figure 4, we have selected velocities as the outputs (in order to compare our data to real neural tuning curves) and we have represented those velocities as the currents ($I_1,...I_n$). The original analysis of Zwislocki's tapered transmission line model did not produce the steep high frequency band edges that are observed in real cochleas. This deficiency was a major driving force behind the early development of more complex cochlear models. Recently, it was found that the original analysis placed the Zwislocki model in an inappropriate mode of operation (Lewis, 1984). In this mode, determined by the relative parameter values, the high frequency band edges had very gentle slopes. Solving the partial differential equations for the model with the help of a commonly used simplification (the WKB approximation), one finds a second mode of operation. In this mode, the model produces all five of the properties that we desire, including extraordinarily steep high-frequency band edges.

# RESULTS

We were curious to know whether or not the newly-found mode of operation, with its very steep high-frequency band edges, could be found in a finite-element version of the model. If so, we should be able to realize a lumped, analog version of the Zwislocki model for use as a practical front-end filter for acoustical image formation and processing. We decided to implement the finite element model in SPICE. SPICE is a software package that is used for electrical circuit simulation. Our SPICE model showed the following: As long as the model was made up of enough segments, and as long as the elements had appropriate parameter values, the second mode of operation indeed was available. Furthermore, it was the predominant mode of operation when the parameter values of the model were matched to biophysical data for the cochlea.

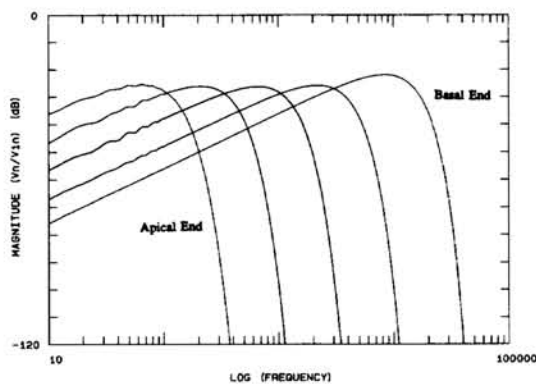

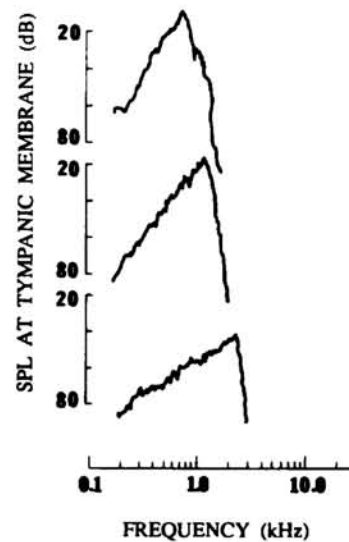

Figure 5 - Frequency response of the basilar membrane velocity.

Figure 6 - Inverted neural tuning curves from three afferent fibers of a cat cochlea (Kiang and Moxon, 1974).

Figure 5 shows the magnitude of the electrical analog's response plotted against frequency on log-log coordinates. The five curves correspond to five different locations along our model. The cutoff frequencies span approximately seven octaves. Further adjustments of the parameters will be needed in order to shift these curves to span the lower seven octaves of human audition. For comparison, figure 6 shows threshold response curves of a cat cochlea from a paper by Kiang and Moxon (1974). These curves are inverted intentionally because Kiang and Moxon plotted stimulus threshold vs. frequency rather than response amplitude vs. frequency. We use these neural tuning curves for comparison because direct observations of cochlear mechanics have been limited to the basal end. Furthermore, in the realm of single frequencies and small signals, Evans has produced compelling evidence that this is a valid comparison (Evans, in press). These three curves are typical of the lower seven octaves of hearing. One obvious discrepancy between Kiang and Moxon's data and our results is that our model does not exhibit the sharp corners occurring at the band edges. The term *sharp corner* denotes the fact that the transition between the shallow rising edge and steep falling edge of a given curve is abrupt i.e. the corner is not rounded.

Figure 7 shows what happens to the response curve at a single location along our model as the number of stages is increased. The curve on the right, in figure 7, was derived with 500 stages and does not change much as we increase the number of stages indefinitely. Thus the curve represents a convergence of the solution of the lumped parameter Zwislocki model to the distributed parameter model. The middle curve was derived with 100 stages and the left-hand curve was derived with 50. In any lumped-element transmission line, there occurs an artifactual cutoff which occurs roughly at the point where the given input wavelength exceeds the dimensions of the lumped elements. If we do not lump the stages in our model finely enough, we observe this artifactual cutoff as opposed to the *true* cutoff of Zwislocki's distributed parameter model. This phenomena is clearly observed in the curve derived from 50 stages and may account for the sharper corners in response curves from real cochleas. However, in order to make our finite element model operate in a manner analogous to that of the distributed parameter Zwislocki model we need approximately 500 stages.

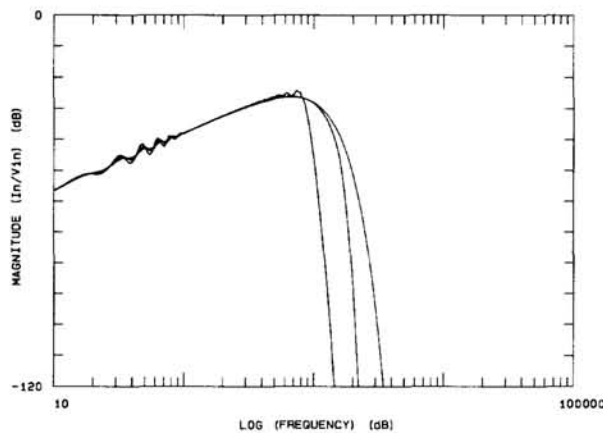

Figure 7 - Convergence of cut-toff points as the number of branches increase.

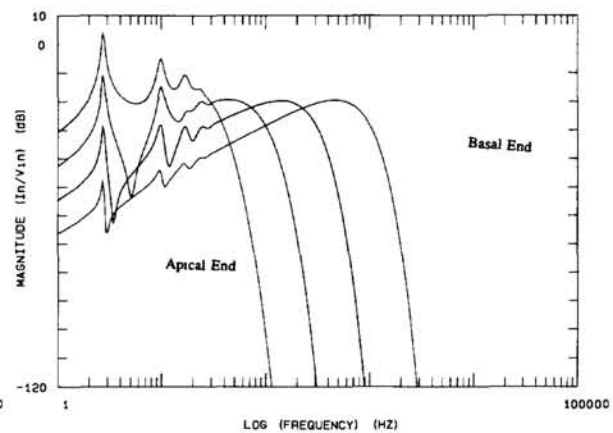

Figure 8 - Frequency response of the basilar membrane velocity without the Heliocotrema.

A critical element in the Zwislocki model is a terminating resistor, representing the heliocotrema (see $R_h$ in figure 3). The heliocotrema is a small hole at the end of the basilar membrane. Figure 8, shows the effects of removing that resistor. The irregular frequency characteristics are quite different from the experimental data and represent possibly wild excursions of the basilar membrane.

Figure 9, shows phase data for the Zwislocki model, which is linear as a function of frequency. Anderson et al (1971), show similar results in the squirrel monkey.

With lumped-element analysis we are able to obtain temporal as well as spectral responses. For a temporal waveform such as an acoustic pulse, the linear relationship between phase and frequency guarantees that those Fourier components which pass through the spectral filter will be reassembled with proper phase relationships at the output of the filter. As it travels down the basilar membrane, the temporal waveform will simply be smoothed, due to loss of its higher-frequency components and delayed, due to the linear phase shift. Figure 10 shows the response of our electrical analog to a 1 msec wide square pulse at the input. The curves represent the time courses of basilar membrane displacement at four equally spaced locations along the cochlea. The curve on the right represents the response at the apical end of the cochlea (the end farthest from the input). The curve on the left represents the response at a point 25 percent of the distance

input end. The impulse responses of mammalian cochleas and of the auditory filters of lower vertebrates all show a slight ringing, again indicating a deficiency in our model.

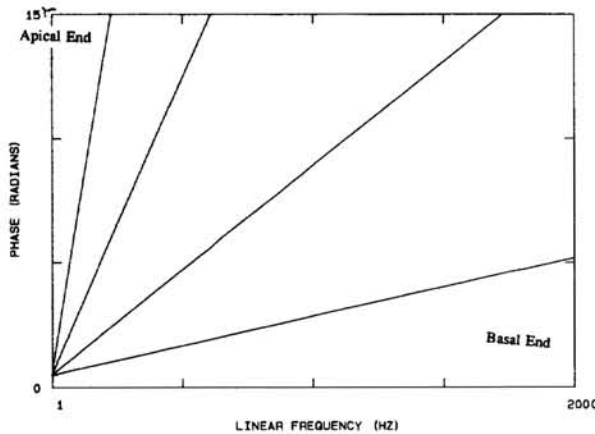

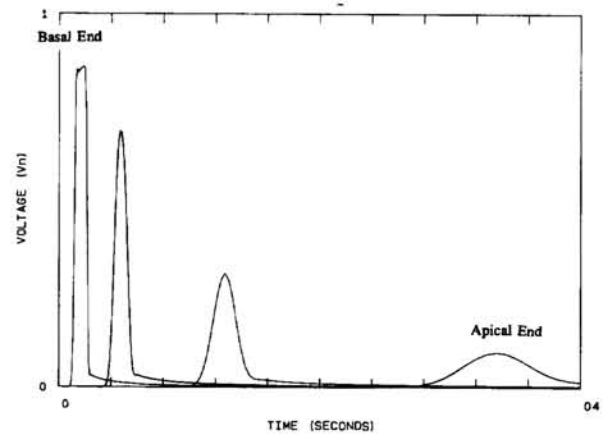

Figure 9 - Phase response of the basilar membrane velocity.

Figure 10 - Traveling square wave pulse along the membrane from the basal to apical end.

## CONCLUSION

Research activity studying the function of higher level brain processing is in its infancy and little is known about how the various features of the cochlea, such as linear phase, sharp band edges, as well as nonlinear features, such as two-tone suppression and cubic difference tone excitation, are used by the brain. Therefore, our approach, in developing a cochlear model, is to incorporate only the most essential ingredients. We have incorporated the five properties mentioned in the introduction which provide simplicity of analysis, economy of hardware construction, and the preservation of both temporal and spectral resolution. The inclusion of these properties is also consistent with the fact that they are found in numerous species.

We have found that in the correct mode of operation a tapered transmission line model can exhibit these five important cochlear properties. A lumped-element approximation can be used to simulate this model as long as at least 500 stages are used. As observed in figure 7, by decreasing the number of stages below 500, the solution to the lumped-element model no longer adheres to the Zwislocki model. In fact, the output of the coursely lumped model more closely resembles the neural tuning data of the cochlea in that it produces very sharp corners. There is some evidence that indicates the cochlea is constructed of discrete components. Indeed, the hair cells themselves are discretized. If this idea is valid, a model constructed of as little as 50 branches may more accurately represent the cochlear mechanics then the Zwislocki model.

Our simple model has some drawbacks in that it does not replicate various properties of the cochlea. For example, it does not span the full ten octaves of human audition, nor does it explain any of the experimentally observed nonlinear aspects seen in the cochlea. However, we take this approach because it provides us with a powerful analysis tool that will enable us to study the behavior of lumped-element cochlear models. This tool will allow us to proceed to the next step; the building of a hardware analog of the cochlea.

# RESEARCH DIRECTIONS

In and of itself, the tapered shared element travelling wave structure we have chosen is interesting to analyze. In order to get even further insight into how this filter works and to aid in the building of a hardware version of such a filter, we plan to study the placement of the poles and zeroes of the transfer function at each tap along the structure. In a travelling wave transmission line we expect that the transfer function at each tap will have the same denominator. Therefore, it must be the numerators of the transfer functions which will change, i.e. the zeroes will change from tap to tap. It will be of interest to see what role the zeroes play in such a ladder structure. Furthermore, it will be of great interest to us to study what happens to the poles and zeroes of the transfer function at each tap as the number of stages is increased (approaching the distributed parameter filter), or decreased (approaching the lumped-element cutoff version of the filter with sharper corners). We should emphasize that our circuit is bi-directional, i.e. there is loading from the stages before and after each tap, as in the real cochlea. For this reason, we must consider carefully the options for hardware realization of our circuit. We might choose to make a mechanical structure on silicon or some other medium, or we could convert our structure into a uni-directional circuit and build it as a digital or analog circuit.

Using this design we plan to build an acoustic imaging device that will enable us to explore various signal processing tasks. One such task would be to extract acoustic signals from noise.

All species need to cope with two types of noise, internal sensor and amplifier noise, and external noise such as that generated by wind. Spectral decomposition is on effective way to deal with internal noise. For example, the amplitudes of the spectral components in the passband of a filter are largely undiminished, whereas the broadband noise, passed by the filter, is proportional to the square root of the bandwidth. External noise reduction can be accomplished by spatial decomposition. When temporal resolution is preserved in signals, spatial decomposition can be achieved by cross correlation of the signals from two ears. Therefore, from these two properties, spectral and temporal resolution, one can construct an acoustic imaging system in which signals buried in a sea of noise can be extracted.

## Acknowledgments

We would like to thank Thuan Nguyen for figure 1, Eva Poinar who helped with the figures, Michael Sneary for valuable discussion, and Bruce Parnas for help with programming.

## References

Anderson, D.J., Rose, J.E., Hind, J.E., Brugge J.F., Temporal Position of Discharges in Single Auditory Nerve Fibers Within the Cycle of a Sine-Wave Stimulus: Frequency and Intensity Effects, *J. Acoust. Soc. Am..*, 49, 1131-1139, 1971.

Evans, E.F., Cochlear Filtering: A View Seen Through the Temporal Discharge Patterns of Single Cochlear Nerve Fibers. A talk given at the 1988 NATO advanced workshop, to be published as (J.P. Wilson, D.T. Kemp, eds.) Mechanics of Hearing, Plenum Press, N.Y.

Kiang, N.Y.S., Moxon, E.C., Tails of Tuning Curves of Auditory-Nerve Fibers, *J. Acoust. Soc. Am.*, 55, 620-630, 1974.

Lewis, E.R., High Frequency Rolloff in a Cochlear Model Without critical-layer resonance, *J. Acoust. Soc. Am.*, 76 (3) September, 1984.

Lyon, R.F., Mead, C.A., An Analog Electronic Cochlea, *IEEE Trans.-ASSP*, 36, 1119-1134, 1988.

Zweig, G., Lipes, R., Pierce, J.R., The Cochlear Compromise, *J. Acoust. Soc. Am.*, 59, 975-982, 1976.

Zwislocki, J., Analysis of Some Auditory Characteristics, in Handbook of Mathematical Psychology, Vol. 3, (Wiley, New York), pp. 1-97, 1965.